# A Massively Parallel Digital

# Learning Processor

**Hans Peter Graf**          **Srihari Cadambi**          **Igor Durdanovic**
*hpg@nec-labs.com*        *cadambi@nec-labs.com*        *igord@nec-labs.com*

**Venkata Jakkula**    **Murugan Sankardadass**    **Eric Cosatto**    **Srimat Chakradhar**
*Jakkula@nec-labs.com  murugs@nec-labs.com   cosatto@nec-labs.com chak@nec-labs.com*

NEC Laboratories, America
4 Independence Way, Suite 200;  Princeton, NJ 07738, USA

## Abstract

We present a new, massively parallel architecture for accelerating machine learning algorithms, based on arrays of vector processing elements (VPEs) with variable-resolution arithmetic. Groups of VPEs operate in SIMD (single instruction multiple data) mode, and each group is connected to an independent memory bank. The memory bandwidth thus scales with the number of VPEs, while the main data flows are local, keeping power dissipation low. With 256 VPEs, implemented on two FPGAs (field programmable gate array) chips, we obtain a sustained speed of 19 GMACS (billion multiply-accumulate per sec.) for SVM training, and 86 GMACS for SVM classification. This performance is more than an order of magnitude higher than that of any FPGA implementation reported so far. The speed on one FPGA is similar to the fastest speeds published on a Graphics Processor for the MNIST problem, despite a clock rate that is an order of magnitude lower. Tests with Convolutional Neural Networks show similar compute performances. This massively parallel architecture is particularly attractive for embedded applications, where low power dissipation is critical.

## 1      Introduction

Machine learning demands higher and higher compute-performance, but serial processors are not improving that much anymore - at least not as quickly as they used to. Mainstream processor development is moving to multi-core systems, using shared memory technology to hide the parallel nature of the processors. But shared memory technology does not scale to hundreds or thousands of cores. In order to reach such levels of parallelization alternative approaches have to be developed. Massively parallel general-purpose computers had limited success so far, because of difficulties programming these machines, and they remain a niche market, mostly in high-performance computing. Yet processors specialized for certain application domains, such as graphics processors or routing processors[1], have been parallelized to several hundred cores and are successful mass products. They improve performance over general-purpose processors by focusing on a few key algorithmic elements, yet still maintain enough flexibility that they can be programmed for a variety of applications. We explore in this paper if a similar approach can lead to efficient machine learning processors.

Several processors optimized for machine learning, in particular for neural networks, were developed during the 1980's and 90's. Examples are the Synapse-1 architecture [1], or the Connectionist Network Supercomputer, CNS1 [2]. Recently there has been less activity in this field, but some accelerators are sold today for specific applications, such as the Axeon [3] processor for power train control of cars. Beside digital processors a large number of analog circuits were built, emulating neural network structures. Extremely high performance with low power dissipation is achievable, see e.g. [4][5], but these networks have little flexibility. SVM implementations on FPGA have been demonstrated in recent years [6-8], yet reached only low compute-performances. All machine learning processors had only limited success so far, indicating how difficult it is to find a good combination of performance, flexibility, price and ease of use. An important consideration is that many applications of machine learning, such as video analysis, data mining, or personalization of services, show the most promise in embedded systems. Embedded learning requires high compute performance while dissipating little power, a combination that is difficult to achieve, and so far required application specific IC (ASIC). Our aim is to develop architectures that meet the requirements for embedded learning, but are programmable and therefore can be used in a wide range of applications.

With the goal of analyzing different architectures we designed a development and testing environment where the parallel computation is mapped onto FPGA's. Initially this system was intended only for experimentation, but its performance is so high that this platform is useful in its own right as accelerator for high-performance systems. While the experiments shown here emphasize high performance, the architecture has been designed from the start for low power dissipation. The main features for achieving this goal are: low-resolution arithmetic, keeping the main data flow local, low operating frequencies, and a modular design, so that unused parts can be powered down dynamically. All results shown here are from the test platform; migration to low-power FPGA or chip designs are done in a later stage.

## 2 Algorithms - Arithmetic - Architecture

For a substantial improvement over a general purpose processor, the algorithms, the arithmetic units, as well as the architecture have to be optimized simultaneously. This is not just an exercise in hardware design, but algorithms and their software implementations have to be developed concurrently. Most machine learning algorithms have not been developed with parallelization in mind. Therefore, we first need to find good parallel versions, identify their performance bottlenecks, and then extract common computational patterns that can be mapped into accelerator hardware.

### 2.1 Algorithms

Characteristic for machine learning is that large amounts of data need to be processed, often with predictable data access patterns and no dependency between operations over large segments of the computation. This is why data-parallelization can often provide good accelerations on multi-core chips, clusters of machines, or even on loosely coupled networks of machines. Using MapReduce, speedups linear with the number of processors have been reported in [9] for several machine learning algorithms. Up to 16 cores were tested, and simulations indicate good scaling to more processors in some cases.

Many algorithms, such as KNN, K-means clustering, LVQ, and Neural Networks can be reduced to forms where the computation is dominated by vector-matrix multiplications, which are easily parallelizable. For Convolutional Neural Networks (CNN) the data flow can be complex, yet the core of the computation is a convolution, an operation which has been studied extensively for parallel implementations. For Support Vector Machines (SVM), several parallel algorithms were described, but most saturate quickly for more than 16 processors. Scaling to larger numbers of processors has been demonstrated, applying MapReduce on a graphics processor with 128 cores [10]. Another implementation on a cluster of 48 dual-core machines (with 384 MMX units) [11] scales even super-linearly, and, according to simulations, scales to thousands of cores.

Based on this analysis it is clear that vector-matrix and matrix-matrix multiplications for large vector dimensionalities and large numbers of vectors must be handled efficiently. Yet this alone is

not sufficient since data access patterns vary greatly between algorithms. We analyze this here in more detail for SVM and CNN. These algorithms were chosen, because they are widely used for industrial applications and cover a broad range of computation, I/O, and memory requirements.

The characteristics of the SVM training are summarized in Table 1. We use an approach similar to the one described in [11] to split different parts of the computation between a host CPU and the FPGA accelerator. For large dimensions $d$ of the vectors the calculation of the columns of the kernel matrix dominates by far. This is needed to update the gradients, and in the present implementation, only this part is mapped onto the FPGA. If the dimensionality $d$ is smaller than around 100, operations 2 and 5 can become bottlenecks and should also be mapped onto the accelerator. Challenging is that for each kernel computation a new data vector has to be loaded into the processor, leading to very high I/O requirements. We consider here dimensions of $10 - 10^4$ and numbers of training data of $10^5 - 10^7$, resulting easily in Gigabytes that need to be transferred to the processors at each iteration.

| | Operation | Computation | IO | Unit |
|---|---|---|---|---|
| 1 | Initialize all $\alpha_x$, $G_x$ | 2n | 2n | CPU |
| - | Do | | | |
| 2 | Find working set $\alpha_j$, $\alpha_i$ | I * 2n | I * 2n | CPU |
| 3 | Update $\alpha_j$, $\alpha_i$ | I * 10 | I * 2 | CPU |
| 4 | Get 2 columns of kernel matrix | I * 2nd | I * (2d+2dn) | FPGA |
| 5 | Update gradients $G_x$ | I * n | I * n | CPU |
| 6 | While not converged | | | |

**Table 1:** *Compute- and IO-requirements of each step for SVM training (SMO algorithm). n: number of training data; d: dimension of the vectors; G: gradients; $\alpha$: support vector factors; I: number of iterations. The last column indicates whether the execution happens on the host CPU or the accelerator FPGA. It is assumed that the kernel computation requires a dot product between vectors (e.g. rbf, polynomial, tanh kernels).*

Neural network algorithms are essentially sequences of vector-matrix multiplications, but networks with special connectivity patterns, such as convolutional networks have very different IO characteristics than fully connected networks. Table 2 shows the computation and IO requirements for scanning several convolution kernels over one input plane. A full network requires multiple of these operations for one layer, with nonlinearities between layers. We map all operations onto the FPGA accelerator, since intermediate results are re-used right away. The most significant difference to between the SVM and CNN is the Compute/IO ratio: SVM: ~ 1; CNN: ~ $L*k^2 > 100$. Therefore the requirements for these two algorithms are very different, and handling both cases efficiently is quite a challenge for an architecture design.

| | Operation | Computation | IO | Unit |
|---|---|---|---|---|
| 1 | Load L kernels | | $L* k^2$ | FPGA |
| | For all input pixels | | | FPGA |
| 2 | Shift in new pixel | | n* m | FPGA |
| 3 | Multiply kernels | $n * m * L * k^2$ | | FPGA |
| 4 | Shift out result | | n*m | FPGA |

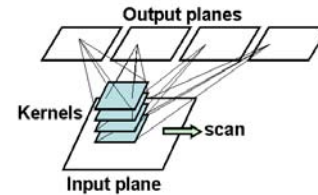

**Table 2:** *Compute- and IO-requirements for CNN computation (forward pass), where l kernels of size k*k are scanned simultaneously over an input plane of size n*m. This is representative for implementations with kernel unrolling (kernel pixels processed in parallel). Internal shifts, computation of the non-linearity, and border effects not shown.*

## 2.2   Arithmetic

Hardware can be built much more compactly and runs with lower power dissipation, if it uses fixed-point instead of floating-point operations. Fortunately, many learning algorithms tolerate a low resolution in most of the computations. This has been investigated extensively for neural

networks [12][13], but less so for other learning algorithms. Learning from data is inherently a noisy process, because we see only a sparse sampling of the true probability distributions. A different type of noise is introduced in gradient descent algorithms, when only a few training data are used at a time to move the optimization forward iteratively. This noise is particularly pronounced for stochastic gradient descent. There is no point in representing noisy variables with high resolution, and it is therefore a property inherent to many algorithms that low-resolution computation can be used.

It is important, not to confuse this tolerance to low resolution with the resolution required to avoid numeric instabilities. Some of the computations have to be performed with a high resolution, in particular for variables that are updated incrementally. They maintain the state of the optimization and may change in very small steps. But usually by far the largest part of the computation can be executed at a low resolution. Key is that the hardware is flexible enough and can take advantage of reduced resolution while handling high resolution where necessary.

| Problem | Kernel: Float | | | Kernel: 16 bit fixed point | | | |
|---------|---------------|---|---|----------------------------|---|---|---|
| | Obj. f. | # SV | F-score | Obj. f. | # SV | F-score | F-sc. (4b in) |
| Adult | 31,930.77 | 11,486 | 77.58 | 31,930.1 | 11,490 | 77.63 | NA |
| Forest | 653,170.7 | 49,333 | 98.29 | 652,758 | 49,299 | 98.28 | NA |
| MNIST | 4,960.13 | 6,172 | 99.12 | 4,959.64 | 6,166 | 99.11 | 99.11 |
| NORB | 1,243.71 | 3,077 | 93.34 | 1,244.76 | 3,154 | 93.26 | 92.78 |

**Table 3:** *Comparison of the results of SVM training when the kernels are represented with floating point numbers (32 or 64 bits) (left half) and with 16 bit fixed point (right half). The last column shows the results when the resolution of the training data is reduced from 8 bit to 4 bit. For NORB this reduces the accuracy; all other differences in accuracy are not significant. All are two class problems: Adult: n=32,562, d=122; Forest: n=522,000, d=54 (2 against the rest); MNIST: n=60,000, d=784 (odd–even); NORB: n=48,560, d=5,184.*

We developed a simulator that allows running the training algorithms with various resolutions in each of the variables. A few examples for SVM training are shown in Table 3. Reducing the resolution of the kernel values from double or float to 16 bit fixed point representations does not affect the accuracy for any of the problems. Therefore all the multiplications in the dot products for the kernel computation can be done in low resolutions (4–16 bit in the factors), but the accumulator needs sufficient resolution to avoid over/under flow (48 bit). Once the calculation of the kernel value is completed, it can be reduced to 16 bit. A low resolution of 16 bit is also tolerable for the α values, but a high resolution is required for the gradients (double). For Neural Networks, including CNN, several studies have confirmed that states and gradients can be kept at low resolutions (<16 bit), but the weights must be maintained at a high resolution (float) (see e.g. [12]). In our own evaluations 24 bits in the weights tend to be sufficient. Once the network is trained, for the classification low resolutions can be used for the weights as well (<16 bit).

## 2.3    Architecture

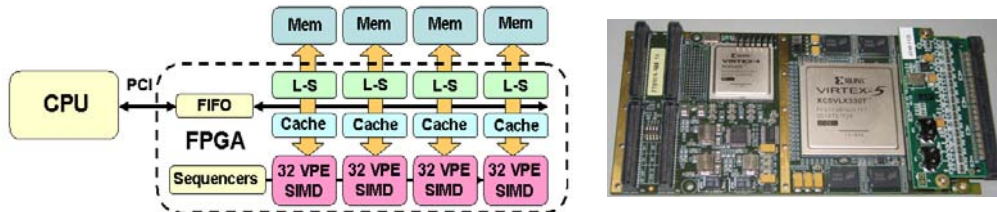

**Figure 1:** Left: *Schematic of the architecture with the main data flows; on one FPGA 128 VPE are configured into four SIMD groups; L-S: Load-store units.* Right: *Picture of an FPGA board; in our experiments one or two of them are used, connected via PCI bus to a host CPU.*

Based on the analysis above, it is clear that the architecture must be optimized for processing massive amounts of data with relatively low precision. Most of the time, data access patterns are predictable and data are processed in blocks that can be stored contiguously. This type of computation is well suited for vector processing, and simple vector processing elements (VPE) with fixed-point arithmetic can handle the operations. Since typically large blocks of data are processed with the same operation, groups of VPE can work in SIMD (single instruction multiple data) mode. Algorithms must then be segmented to map the high-volume, low precision parts onto the vector accelerators and parts requiring high precision arithmetic onto the CPU.

The most important design decision is the organization of the memory. Most memory accesses are done in large blocks, so that the data can be streamed, making complex caching unnecessary. This is fortunate, since the amounts of data to be loaded onto the processor are so large that conventional caching strategies would be overwhelmed anyway. Because the blocks tend to be large, a high data bandwidth is crucial, but latency for starting a block transfer is less critical. Therefore we can use regular DDR memories and still get high IO rates. This led to the design shown schematically in Figure 1, where independent memory banks are connected via separate IO ports for each group of 32 VPE.

By connecting multiple of the units shown in Figure 1 to a CPU, this architecture scales to larger numbers of VPE. Parallel data IO and parallel memory access scale simultaneously with the number of parallel cores, and we therefore refer to this as the $P^3$ (P-cube) architecture. Notice also that the main data flow is only local between a group of VPE and its own memory block. Avoiding movements of data over long distances is crucial for low power dissipation. How far this architecture can reasonably scale with one CPU depends on the algorithms, the amount of data and the vector dimensionality (see below). A few hundred VPE per CPU have provided good accelerations in all our tests, and much higher numbers are possible with multi-core CPUs and faster CPU-FPGA connections.

# 3    Implementation of the $P^3$ Architecture

This architecture fits surprisingly well onto some of the recent FPGA chips that are available with several hundred Digital Signal Processors (DSP) units and over 1,000 IO pins for data transfers. The boards used here contain each one Xilinx Virtex 5 LX330T-2 FPGA coupled to 4 independent DDR2 SDRAM with a total of 1GB, and 2 independent 4MB SSRAM memory banks (commercial board from AlphaData). One FPGA chip contains 192 DSP with a maximum speed of 550MHz, which corresponds to a theoretical compute-performance of 105.6 GMACS (18 bit and 25 bit operands). There is a total of 14 Mbit of on-chip memory, and the chip incorporates 960 pins for data IO. Due to routing overhead, not all DSP units can be used and the actual clock frequencies tend to be considerably lower than what is advertised for such chips (typically 230MHz or less for our designs). Nevertheless, we obtain high performances because we can use a large number of DSP units for executing the main computation.

The main architecture features are:
- *Parallel processing (on one chip):* 128 VPE (hardware DSP) are divided into 4 blocks of 32, each group controlled by one sequencer with a vector instruction set.
- *Custom Precision*: Data are represented with 1 to 16 bit resolution. Higher resolutions are possible by operating multiple DSP as one processor.
- *Overlapping Computation and Communication*: CPU-FPGA communication is overlapped with the FPGA computation.
- *Overlap Memory Operations with Computation:* All loads and stores from the FPGA to off-chip memory are performed concurrently with computations.
- *High Off-chip Memory Bandwidth:* 6 independent data ports, each 32 bits wide, access banked memories concurrently (12GB/s per chip).

- *Streaming Data Flow, Simple Access Patterns:* Load/store units are tailored for streaming input and output data, and for simple, bursty access patterns. Caching is done under application control with dual-port memory on chip.
- *Load/store with (de)compression:* For an increase of effective IO bandwidth the load/store units provide compression and decompression in hardware.

Figure 2 shows the configuration of the VPEs for vector dot product computation used for SVM training and classification. For training, the main computation is the calculation of one column of the kernel matrix. One vector is pre-fetched and stored in on-chip memory. All other vectors are streamed in from off-chip memory banks 1-4. Since this is a regular and predictable access pattern, we can utilize burst-mode, achieving a throughput of close to one memory word per cycle. But the speed is nevertheless IO bound. When several vectors can be stored on-chip, as is the case for classification, then the speed becomes compute-bound.

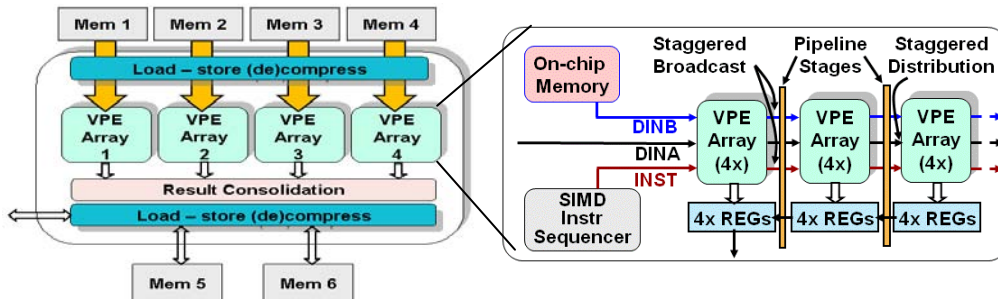

**Figure 2:** *Architecture for vector dot-product computation. The left side shows a high-level schematic with the main data flow. The data are streamed from memory banks 1-4 to the VPE arrays, while memory banks 5 and 6, alternatively receive results or stream them back to the host. The right side shows how a group of VPE is pipelined to improve clock speed.*

The operation for SVM training on the FPGA corresponds to a vector-matrix multiplication and the one for classification to a matrix-matrix multiplication. Therefore the configuration of Figure 2 is useful for many other algorithms as well, where operations with large vectors and matrices are needed, such as Neural Networks. We implemented a specialized configuration for Convolutional Neural Networks, for more efficiency and lower power dissipation. The VPE are daisy-chained and operate as systolic array. In this way we can take advantage of the high computation to IO ratio (Table 2) to reduce the data transfers from memory.

## 4    Evaluations

We evaluated SVM training and classification with the NORB and MNIST problems, the latter with up to 2 million training samples (data from [11]). Both are benchmarks with vectors of high dimensionality, representative for applications in image and video analysis. The computation is split between CPU and FPGA as indicated by Table 1. The DDR2 memory banks are clocked at 230MHz, providing double that rate for data transfers. The data may be compressed to save IO bandwidth. On the FPGA they are decompressed first and distributed to the VPE. In our case, a 32 bit word contains eight 4-bit vector components. Four 32 bit words are needed to feed all 32 VPEs of a group; therefore clocking the VPE faster than 115MHz does not improve performance. A VPE executes a multiplication plus add operation in one clock cycle, resulting in a theoretical maximum of 14.7 GMACS per chip. The sustained compute-rate is lower, about 9.4 GMACS, due to overhead (see Table 4). The computation on the host CPU overlaps with that on the FPGA, and has no effect on the speed in the experiments shown here. For the classification the VPE can be clocked higher, at 230 MHz. By using 4-bit operands we can execute 2 multiply-accumulates simultaneously on one DSP, resulting in speed that is more than four times higher and a sustained 43.0 GMACS limited by the number and speed of the VPE. Adding a second FPGA card doubles the speed, showing little saturation effects yet, but for more FPGA per CPU there will be saturation (see Fig. 3). The compute speed in GMACS obtained for NORB is almost identical.

| # | Iterations | CPU | | CPU+MMX | | CPU+FPGA | | CPU+2 FPGA | |
|---|---|---|---|---|---|---|---|---|---|
| | | time | speed | time | speed | time | speed | time | speed |
| 60k | 8,000 | 754s | 0.5 | 240 s | 1.57 | 40 s | 9.42 | 21 s | 17.9 |
| 2M | 266,900 | -- | -- | 531,534 s | 1.58 | 88,589 s | 9.48 | 48,723 s | 17.2 |

**Table 4:** *Training times and average compute speed for SVM training. Systems tested: CPU, Opteron, 2.2GHz; CPU using MMX; CPU with one FPGA; CPU with two FPGA boards. Results are shown for training sizes of 60k and 2M samples. Compute speed is in GMACS (just kernel computations). Training algorithm: SMO with second order working set selection.*

Parallelizations of SVM training have been reported recently for a GPU [10] and for a cluster [11], both using the MNIST data. In [10] different bounds for stopping were used than here and in [11]. Nevertheless, a comparison of the compute performance is possible, because based on the number of iterations we can compute the average GMACS for the kernel computations. As can be seen in Table 5 a single FPGA is similar in speed to a GPU with 128 stream processors, despite a clock rate that is about 5.5 times lower for I/O and 11 times lower for the VPE. The cluster with 384 MMX units is about 6 times faster than one FPGA with 128 VPE, but dissipates about two orders of magnitude more electric power. For the FPGA this calculation includes only the computation of the kernel values while the part on the CPU is neglected. This is justified for this study, because the rest of the calculations can be mapped on the FPGA as well and will increase the power dissipation only minimally.

| Processor | Number of cores | Clock speed | Operand type | Power dissipation | Average compute speed |
|---|---|---|---|---|---|
| CPU (Opteron) | 1 | 2.2 GHz | float | 40 W | 0.5 GMACS |
| GPU (from [10]) | 128 | 1.35 GHz | float | 80 W | 7.4 GMACS |
| Cluster (from [11]) | 384 | 1.6 GHz | byte | > 1 kW | 54 GMACS |
| FPGA | 128 | 0.12 GHz | 4 bit nibble | 9 W | 9.4 GMACS |

Table 5: *Comparison of performances for SVM training (MNIST data). GPU: Nvidia 8800 GTX. Cluster: 48 dual core CPU (Athlon), 384 MMX units. The GPU was training with 60k samples ([10], table 2, second order), the cluster trained with 2 million samples.*

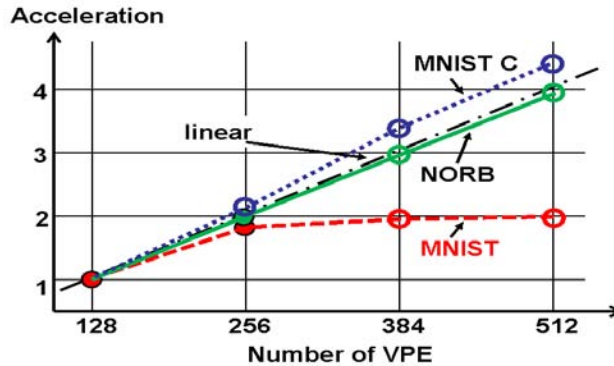

**Figure 3:** *Acceleration of SVM training as a function of the number of VPE. MNIST n: 2,000,000, d=784; NORB: n=48,560, d=5,184. The points for 128 and 256 VPE are experimental, the higher ones are simulations. Curves MNIST, NORB: Multiple FPGA are attached to one CPU. Curve MNIST C: Each FPGA is attached to a separate host CPU.*

Scaling of the acceleration with the number of VPEs is shown in Figure 3. The reference speed is that of one FPGA attached to a CPU. The evaluation has been done experimentally for 128 and 256 VPEs, and beyond that with a simulator. The onset of saturation depends on the dimensionality of the vectors, but to a much lesser extent on the number of training vectors (up to the limit of the memory on the FPGA card). MNIST saturates for more than two FPGAs because

then the CPU and FPGA computation times become comparable. For the larger vectors of NORB ($d$=5,184) this saturation starts to be noticeable for more than 4 FPGA. Alternatively, a system can be scaled by grouping multiple CPU, each with one attached FPGA accelerator. Then the scaling follows a linear or even super-linear acceleration (MNIST C) to several thousand VPE. If the CPUs are working in a cluster arrangement, the scaling is similar to the one described in [11].

For convolutional neural networks, the architecture of Figure 2 is modified to allow a block of VPE to operate as systolic array. In this way convolutions can be implemented with minimal data movements. In addition to the convolution, also sub-sampling and non-linear functions plus the logistics to handle multiple layers with arbitrary numbers of kernels in each layer are done on the FPGA. Four separate blocks of such convolvers are packed onto one FPGA, using 100 VPE. Clocked at 115MHz, this architecture provides a maximum of 11.5 GMACS. Including all the overhead the sustained speed is about 10 GMACS.

# 5    Conclusions

By systematically exploiting characteristic properties of machine learning algorithms, we developed a new massively parallel processor architecture that is very efficient and can be scaled to thousands of processing elements. The implementation demonstrated here is more than an order of magnitude higher in performance than previous FPGA implementations of SVM or CNN. For the MNIST problem it is comparable to the fastest GPU implementations reported so far. These results underline the importance of flexibility over raw compute-speed for massively parallel systems. The flexibility of the FPGA allows more efficient routing and packing of the data and the use of computations with the lowest resolution an algorithm permits. The results of Table 5 indicate the potential of this architecture for low-power operation in embedded applications.

## Footnotes

[1] e.g. Nvidia, Quadro FX 5600 graphics processor; Cisco, CRS-1 routing processor

# References

[1] Ramacher, et al. (1995) Synapse-1: A high-speed general purpose parallel neurocomputer system. In *Proc. 9th Intl. Symposium on Parallel Processing (IPPS'95),* pp. 774-781.

[2] Asanovic, K., Beck, Feldman, J., Morgan, N. & Wawrzynek, J. (1994) A Supercomputer for Neural Computation, *Proc. IEEE Intl. Joint Conference on Neural Networks*, pp. 5-9, Orlando, Florida.

[3] Neil, P., (2005) Combining hardware with a powerful automotive MCU for powertrain applications. In *Industrial Embedded Resource Guide*, p. 88.

[4] Korekado, et al. (2003) A Convolutional Neural Network VLSI for Image Recognition Using Merged/Mixed Analog-Digital Architecture, in Proc. 7th KES 2003, Oxford, pp 169-176.

[5] Murasaki, M., Arima, Y. & Shinohara, H. (1993) A 20 Tera-CPS Analog Neural Network Board. In *Proc. Int. Joint Conf. Neural Networks*, pp. 3027 – 3030.

[6] Pedersen, R., Schoeberl, M. (2006), An Embedded Support Vector Machine, WISE 2006.

[7] Dey, S., Kedia, M. Agarwal, N., Basu, A., Embedded Support Vector Machine: Architectural Enhancements and Evaluation, in Proc 20th Int. Conf. VLSI Design.

[8] Anguita, D., Boni, A., Ridella, S., (2003) A Digital Architecture for Support Vector Machines: Theory, Algorithm, and FPGA Implementation, IEEE Trans. Neural Networks, 14/5, pp.993-1009.

[9] Chu, C., Kim, S., Lin, Y., Yu, Y., Bradski, G., Ng, A. & Olukotun, K. (2007) Map-Reduce for Machine Learning on Multicore, *Advances in Neural Information Processing Systems 19*, MIT Press.

[10] Catanzaro, B., Sundaram, N., & Keutzer, K. (2008) Fast Support Vector Machine Training and Classification on Graphics Processors, Proc. 25th Int. Conf. Machine Learning, pp 104-111.

[11] Durdanovic, I., Cosatto, E. & Graf, H. (2007) Large Scale Parallel SVM Implementation. In L. Bottou, O. Chapelle, D. DeCoste, J. Weston (eds.), *Large Scale Kernel Machines,* pp. 105-138, MIT Press.

[12] Simard, P & Graf, H. (1994) Backpropagation without Multiplication. In J. Cowan, G. Tesauro, J. Alspector, (eds.), *Neural Information Processing Systems 6*, pp. 232 – 239, Morgan Kaufmann.

[13] Savich, A., Moussa, M., Areibi, S., (2007) The Impact of Arithmetic Representation on Implementing MLP-BP on FPGAs: A Study, IEEE Trans. Neural Networks, 18/1, pp. 240-252.
